# Deep Learning of Invariant Features via Simulated Fixations in Video

**Will Y. Zou**[1], **Shenghuo Zhu**[3], **Andrew Y. Ng**[2], **Kai Yu**[3]
[1]Department of Electrical Engineering, Stanford University, CA
[2]Department of Computer Science, Stanford University, CA
[3]NEC Laboratories America, Inc., Cupertino, CA
{wzou, ang}@cs.stanford.edu  {zsh, kyu}@sv.nec-labs.com

## Abstract

We apply salient feature detection and tracking in videos to simulate fixations and smooth pursuit in human vision. With tracked sequences as input, a hierarchical network of modules learns invariant features using a temporal slowness constraint. The network encodes invariance which are increasingly complex with hierarchy. Although learned from videos, our features are spatial instead of spatial-temporal, and well suited for extracting features from still images. We applied our features to four datasets (COIL-100, Caltech 101, STL-10, PubFig), and observe a consistent improvement of 4% to 5% in classification accuracy. With this approach, we achieve state-of-the-art recognition accuracy 61% on STL-10 dataset.

## 1 Introduction

Our visual systems are amazingly competent at recognizing patterns in images. During their development, training stimuli are not incoherent sequences of images, but natural visual streams modulated by fixations [1]. Likewise, we expect a machine vision system to learn from coherent image sequences extracted from the natural environment. Through this learning process, it is desired that features become robust to temporal transfromations and perform significantly better in recognition. In this paper, we build an unsupervised deep learning system which exhibits theses properties, thus achieving competitive performance on concrete computer vision benchmarks.

As a learning principle, sparsity is essential to understanding the statistics of natural images [2]. However, it remains unclear to what extent sparsity and subspace pooling [3, 4] could produce invariance exhibited in higher levels of visual systems. Another approach to learning invariance is temporal slowness [1, 5, 6, 7]. Experimental evidence suggests that high-level visual representations become slow-changing and tolerant towards non-trivial transformations, by associating low-level features which appear in a coherent sequence [5].

To learn features using slowness, a key observation is that during our visual fixations, moving objects remain in visual focus for a sustained amount of time through smooth pursuit eye movements. This mechanism ensures that the same object remains in visual exposure, avoiding rapid switching or translations. Simulation for such a mechanism forms an essential part of our proposal. In natural videos, we use spatial-temporal feature detectors to simulate fixations on salient features. At these feature locations, we apply local contrast normalization [8], template matching [9] to find local correspondences between successive video frames. This approach produces training sequences for our unsupervised algorithm. As shown in Figure 1, training input to the neural network is free from abrupt changes but contain non-trivial motion transformations.

In prior work [10, 11, 12], a single layer of features learned using temporal slowness results in translation-invariant edge detectors, reminiscent of complex-cells. However, it remains unclear whether higher levels of invariances [1], such as ones exhibited in IT, can be learned using temporal

Fixed location video sequences      Video sequences obtained with tracking

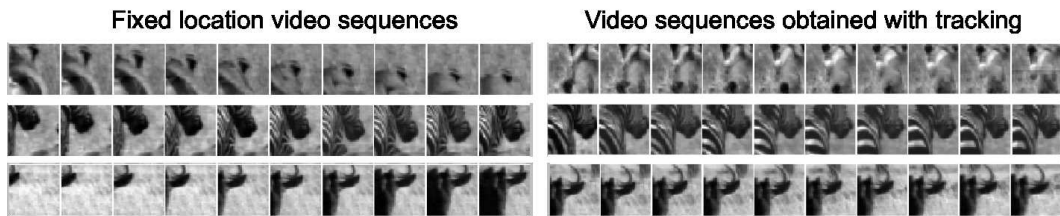

Figure 1: Simulating smooth pursuit eye movements. (Left) Sequences extracted from fixed spatial locations in a video. (Right) Sequences produced by our tracking algorithm.

slowness. In this paper, we focus on developing algorithms that capture higher levels of invariance, by learning multiple layers of representations. By stacking learning modules, we are able to learn features that are increasingly invariant. Using temporal slowness, the first layer units become locally translational invariant, similar to subspace or spatial pooling; the second layer units can then encode more complex invariances such as out-of-plane transformations and non-linear warping.

Using this approach, we show a surprising result that despite being trained on videos, our features encode complex invariances which translate to recognition performance on still images. We carry out our experiments using the self-taught learning framework [13]. We first learn a set of features using simulated fixations in unlabeled videos, and then apply the learned features to classification tasks. The learned features improve accuracy by a significant 4% to 5% across four still image recognition datasets. In particular, we show best classification results to date 61% on the STL-10 [14] dataset. Finally, we quantify the invariance learned using temporal slowness and simulated fixations by a set of control experiments.

## 2    Related work

Unsupervised learning image features from pixels is a relatively new approach in computer vision. Nevertheless, there have been successful application of unsupervised learning algorithms such as Sparse Coding [15, 16], Independent Component Analysis [17], even clustering algorithms [14] on a convincing range of datasets. These algorithms often use such principles as sparsity and feature orthogonality to learn good representations.

Recent work in deep learning such as Le et. al. [18] showed promising results for the application of deep learning to vision. At the same time, these advances suggest challenges for learning deeper layers [19] using purely unsupervised learning. Mobahi et. al. [20] showed that temporal slowness could improve recognition on a video-like COIL-100 dataset. Despite being one of the first to apply temporal slowness in deep architectures, the authors trained a fully supervised convolutional network and used temporal slowness as a regularizing step in the optimization procedure. The influential work of Slow Feature Analysis (SFA) [7] was an early example of unsupervised algorithm using temporal slowness. SFA solves a constrained problem and optimizes for temporal slowness by mapping data into a quadratic expansion and performing eigenvector decomposition. Despite its elegance, SFA's non-linear (quadratic) expansion is slow computationally when applied to high dimensional data. Applications of SFA to computer vision have had limited success, applied primarily to artificially generated graphics data [21]. Bergstra et. al. [12] proposed to train deep architectures with temporal slowness and decorrelation, and illustrated training a first layer on MNIST digits. [22, 23] proposed bi-linear models to represents natural images using a factorial code. Cadieu et. al. [24] trained a two-layer algorithm to learn visual transformations in videos, with limited emphasis on temporal slowness.

The computer vision literature has a number of works which, similar to us, use the idea of video tracking to learn invariant features. Stavens et. al. [25] show improvement in performance when SIFT/HOG parameters are optimized using tracked image patch sequences in specific application domains. Leistner et. al. [26] used natural videos as "weakly supervised" signals to improve random forest classifiers. Lee et. al. [27] introduced video-based descriptors used in hand-held visual recognition systems. In contrast to these recent examples, our algorithm learns features directly from raw image pixels, and adapts to pixel-level image statistics—in particular, it does not rely on hand-designed preprocessing such as SIFT/HOG. Further, since it is implemented by a neural

network, our method can also be used in conjunction with such techniques as fine-tuning with back-propagation. [28, 29]

## 3 Learning Architecture

In this section, we describe the basic modules and the architecture of the learning algorithm. In particular, our learning modules use a combination of temporal slowness and a non-degeneracy principle similar to orthogonality [30, 31]. Each module implements a linear transformation followed by a pooling step. The modules can be activated in a feed-forward manner, making them suitable for forming a deep architecture. To learn invariant features with temporal slowness, we use a two layer network, where the first layer is convolutional and replicates neurons with local receptive field across dense grid locations, and the second (non-convolutional) layer is fully connected.

### 3.1 Learning Module

The input data to our learning module is a coherent sequence of image frames, and all frames in the sequence are indexed by $t$. To learn hidden features $\mathbf{p}^{(t)}$ from data $\mathbf{x}^{(t)}$, the modules are trained by solving the following unconstrained minimization problem:

$$\underset{W}{\text{minimize}} \quad \lambda \sum_{t=1}^{N-1} \|\mathbf{p}^{(t)} - \mathbf{p}^{(t+1)}\|_1 + \sum_{t=1}^{N} \|\mathbf{x}^{(t)} - W^T W \mathbf{x}^{(t)}\|_2^2 \qquad (1)$$

The hidden features $\mathbf{p}^{(t)}$ are mapped from data $\mathbf{x}^{(t)}$ by a feed-forward pass in the network shown Figure 2:

$$\mathbf{p}^{(t)} = \sqrt{H(W\mathbf{x}^{(t)})^2} \qquad (2)$$

This equation describes $L_2$ pooling on a linear network layer. The square and square-root operations are element-wise. This pooling mechanism is implemented by a subspace pooling matrix $H$ with a group size of two [30]. More specifically, each row of $H$ picks and sums two adjacent feature dimensions in a non-overlapping fashion.

The second term in Equation 1 is from the Reconstruction ICA algorithm [31]. It helps avoid degeneracy in the features, and plays a role similar to orthogonalization in Independent Component Analysis [30]. The network encodes the data $\mathbf{x}^{(t)}$ by a matrix-vector multiplication $\mathbf{z}^{(t)} = W\mathbf{x}^{(t)}$, and reconstructs the data with another feed-forward pass $\hat{\mathbf{x}}^{(t)} = W^T \mathbf{z}^{(t)}$. This term can also be interpreted as an auto-encoder reconstruction cost. (See [31] for details.)

Although the algorithm is driven by temporal slowness, sparsity also helps to obtain good features from natural images. Thus, in practice, we further add to Equation 1 an $L_1$-norm sparsity regularization term $\gamma \sum_{t=1}^{N} \|\mathbf{p}^{(t)}\|_1$, to make sure the obtained features have sparse activations.

This basic algorithm trained on the Hans van Hateren's natural video repository [24] produced oriented edge filters. The learned features are highly *invariant to local translations*. The reason for this is that temporal slowness requires hidden features to be slow-changing across time. Using the visualization method of [24], in Figure 3, we vary the interpolation angle in-between pairs of pooled features, and produce a motion of smooth translations. A video of this illustration is also available online.[1]

### 3.2 Stacked Architecture

The first layer modules described in the last section are trained on a smaller patch size (16x16 pixels) of locally tracked video sequences. To construct the set of inputs to the second stacked layer, first layer features are replicated on a dense grid in a larger scale (32x32 pixels). The input to layer two is extracted after $L_2$ pooling. This architecture produces an over-complete number of local 16x16 features across the larger feature area.

The two layer architecture is shown in Figure 4. Due to the high dimensionality of the first layer outputs, we apply PCA to reduce their dimensions for the second layer algorithm. Afterwards,

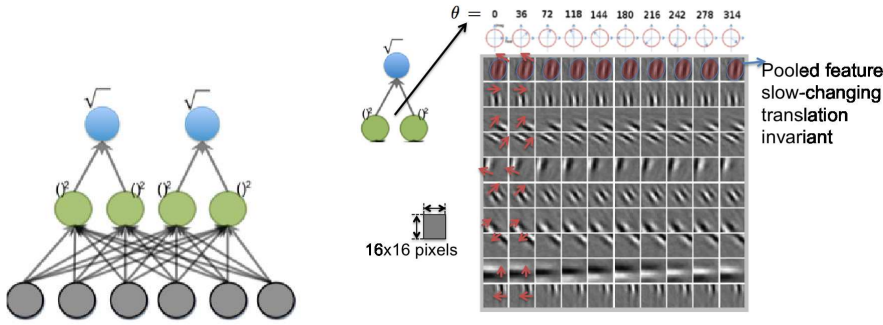

Figure 2: Neural network architecture of the basic learning module

Figure 3: Translational invariance in first layer features; columns correspond to interpolation angle $\theta$ at multiples of 45 degrees

a fully connected module is trained with temporal slowness on the output of PCA. The stacked architecture learns features in a signficantly larger 2-D area than the first layer algorithm, and able to learn invariance to larger-scale transformations seen in videos.

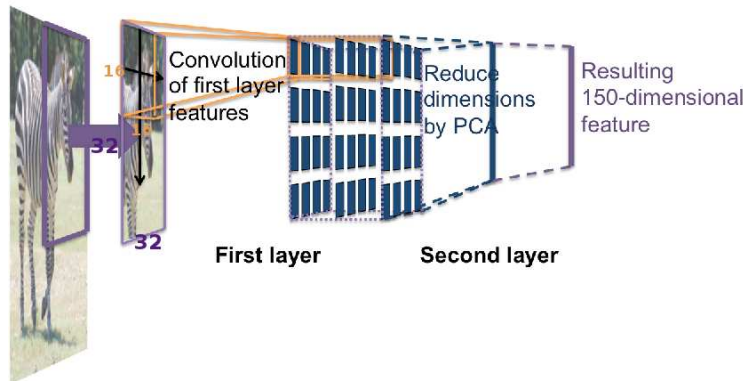

Figure 4: Two-layer architecture of our algorithm used to learn invariance from videos.

### 3.3 Invariance Visualization

After unsupervised training with video sequences, we visualize the features learned by the two layer network. On the left of Figure 5, we show the optimal stimuli which maximally activates each of the first layer pooling units. This is obtained by analytically finding the input that maximizes the output of a pooling unit (subject to the constraint that the input $\mathbf{x}$ has unit norm, $\|\mathbf{x}\|_2 = 1$). The optimal stimuli for units learned without slowness are shown at the top, and appears to give high frequency grating-like patterns. At the bottom, we show the optimal stimuli for features learned with slowness; here, the optimal stimuli appear much smoother because the pairs of Gabor-like features being pooled over are usually a quadrature pair. This implies that the pooling unit is robust to changes in phase positions, which correspond to translations of the Gabor-like feature.

The second layer features are learned on top of the pooled first layer features. We visualize the second layer features by plotting linear combinations of the first layer features' optimal stimuli (as shown on the left of Figure 5), and varying the interpolation angle as in [24]. The result is shown on right of Figure 5, where each row corresponds to the visualization of a single pooling unit. Each row corresponds to a motion sequence to which we would expect the second layer features to be roughly invariant. From this visualization, non-trivial invariances are observed such as non-linear warping, rotation, local non-affine changes and large scale translations. A video animation of this visualization is also available online[2].

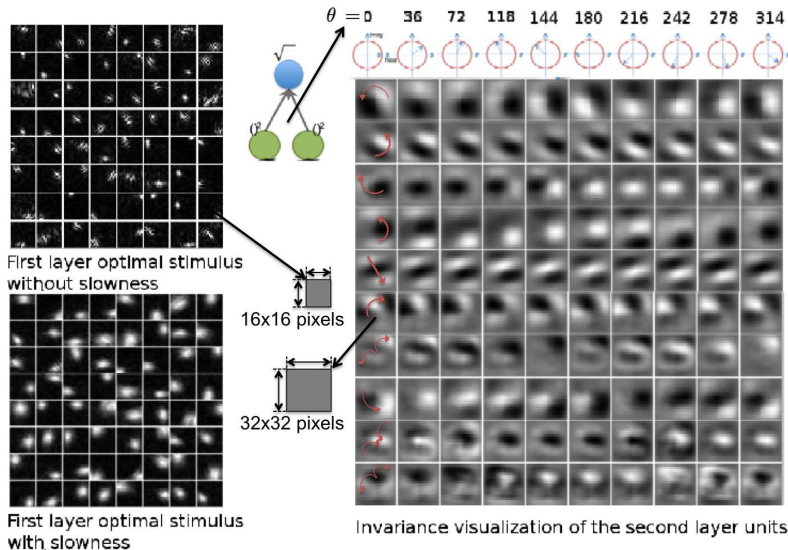

Figure 5: (Left) Comparison of optimal stimuli of *first layer* pooling units (patch size 16x16) learned without (top) and with (bottom) temporal slowness. (Right) visualization of *second layer* features (patch size 32x32), with each row corresponding to one pooling unit. We observe a few non-trivial invariances, such as warping (rows 9 and 10), rotation (first row), local non-affine changes (rows 3, 4, 6, 7), large scale translations (rows 2 and 5).

## 4 Experiments

Our experiments are carried out in a self-taught learning setting [13]. We first train the algorithm on the Hans van Hateren natural scene videos, to learn a set of features. The learned features are then used to classify single images in each of four datasets. Throughout this section, we use gray-scale features to perform recognition.

### 4.1 Training with Tracked Sequences

To extract data from the Hans van Hateren natural video repository, we apply spatial-temporal Difference-of-Gaussian blob detector and select areas of high response to simulate visual fixations. After the initial frame is selected, the image patch is tracked across 20 frames using a tracker we built and customized for this task. The tracker finds local correspondences by calculating Normalized Cross Correlation (NCC) of patches across time which are processed with local contrast normalization.

The first layer algorithm is learned on 16x16 patches with 128 features (pooled from 256 linear bases). The bases are then convolved within the larger 32x32 image patches with a stride of 2. PCA is used to first reduce the dimensions of the response maps to 300 before learning the second layer. The second layer learns 150 features (pooled from 300 linear bases).

### 4.2 Vision Datasets

COIL-100 contains images of 100 objects, each with 72 views. We followed testing protocols in [32, 20]. The videos we trained on to obtain the temporal slowness features were based on the van Hateren videos, and were thus unrelated to COIL-100. The classification experiment is performed on all 100 objects.

In Caltech 101, we followed the common experiment setup given in [33]: we pick 30 images per class as training set, and randomly pick 50 per class (if fewer than 50 left, take the rest) as test set. This is performed 10 times and we report the average classification accuracy.

The STL-10 [34] dataset contains 10 object classes with 5000 training and 8000 test images. There are 10 pre-defined folds of training images, with 500 images in each fold. In each fold, a classifier

Table 1: Acc. COIL-100 (unrelated video)

| Method | Acc. |
|---|---|
| VTU [32] | 79.1% |
| ConvNet regularized with video [20] | 79.77% |
| Our results without video | 82.0% |
| Our results using video | **87.0%** |
| Performance increase by training on video | **+5.0%** |

Table 2: Ave. acc. Caltech 101

| Method | Ave. acc. |
|---|---|
| Two-layer ConvNet [36] | 66.9% |
| ScSPM [37] | 73.2% |
| Hierarchical sparse-coding [38] | 74.0% |
| Macrofeatures [39] | **75.7%** |
| Our results without video | 66.5% |
| Our results using video | 74.6% |
| Performance increase with video | **+8.1%** |

Table 3: Ave. acc. STL-10

| Method | Ave. acc. |
|---|---|
| Reconstruction ICA [31] | 52.9% |
| Sparse Filtering [40] | 53.5% |
| SC features, K-means encoding [16] | 56.0% |
| SC features, SC encoding [16] | 59.0% |
| Local receptive field selection [19] | 60.1% |
| Our result without video | 56.5% |
| Our result using video | **61.0%** |
| Performance increase with video | **+4.5%** |

Table 4: Acc. PubFig faces

| Method | Acc. |
|---|---|
| Our result without video | 86% |
| Our result using video | 90.0% |
| Performance increase with video | **+4.0%** |

is trained on a specific set of 500 training images, and tested on all 8000 testing images. Similar to prior work, the evaluation metric we report is average accuracy across 10 folds. The dataset is suitable for developing unsupervised feature learning and self-taught learning algorithms, since the number of supervised training labels is relatively small.

PubFig [35] is a face recognition dataset with 58,797 images of 200 persons. Face images contain large variation in pose, expression, background and image conditions. Since some of the URL links provided by the authors were broken, we only compare our results using video against our own baseline result without video. 10% of the downloaded data was used as the test set.

### 4.3 Test Pipeline

On still images, we apply our trained network to extract features at dense grid locations. A linear SVM classifier is trained on features from both first and second layers. We did not apply fine-tuning. For COIL-100, we cross validate the average pooling size. A simple four-quadrant pooling is used for STL-10 and PubFig datasets. For Caltech 101, we use a three layer spatial pyramid.

### 4.4 Recognition Results

We report results on COIL-100, Caltech 101, STL-10 and PubFig datasets in tables 1, 2, 3 and 4.

In these experiments, the hyper-parameters are cross-validated. However, performance is not particularly sensitive to the weighting between temporal slowness objective compared to reconstruction objective in Equation 1, as we will illustrate in Section 4.5.2. For each dataset, we compare results using features trained with and without the temporal slowness objective term in Equation 1. Despite the feature being learned from natural videos and then being transferred to different recognition tasks (i.e., self-taught learning), they give excellent performance in our experiments. The application of temporal slowness increases recognition accuracy consistently by 4% to 5%, bringing our results to be competitive with the state-of-the-art.

### 4.5 Control Experiments

#### 4.5.1 Effect of Fixation Simulation and Tracking

We carry out a control experiment to elucidate the difference between features learned using our fixation and smooth pursuit method for extracting video frames (as in Figure 1, right) compared to features learned using non-tracked sequences (Figure 1, left). As shown on the left of Figure 6, training on tracked sequences reduces the translation invariance learned in the second layer. In

comparison to other forms of invariances, translation is less useful because it is easy to encode with spatial pooling [17]. Instead, the features encode other invariance such as different forms of non-linear warping. The advantage of using tracked data is reflected in object recognition performance on the STL-10 dataset. Shown on the right of Figure 6, recognition accuracy is increased by a considerable margin by training on tracked sequences.

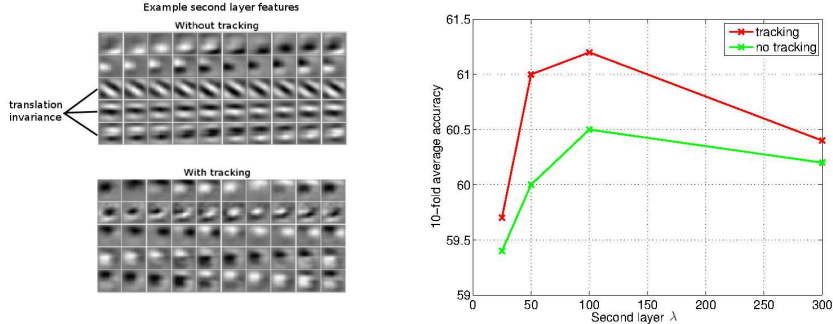

Figure 6: (Left) Comparison of second layer invariance visualization when training data was obtained with tracking and without; (Right) Ave. acc. on STL-10 with features trained on tracked sequences compared to non-tracked; $\lambda$ in this plot is slowness weighting parameter from Equation 1 .

### 4.5.2 Importance of Temporal Slowness to Recognition Performance

To understand how much the slowness principle helps to learn good features, we vary the slowness parameter across a range of values to observe its effect on recognition accuracy. Figure 7 shows recognition accuracy on STL-10, plotted as a function of a slowness weighting parameter $\lambda$ in the first and second layers. On both layers, accuracy increases considerably with $\lambda$, and then levels off slowly as the weighting parameter becomes large. The performance also appears to be reasonably robust to the choice of $\lambda$, so long as the parameter is in the high-value regime.

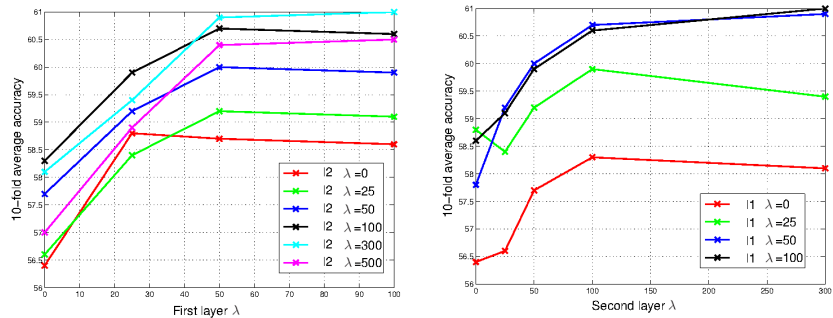

Figure 7: Performance on STL-10 versus the amount of temporal slowness, on the first layer (left) and second layer (right); in these plots $\lambda$ is the slowness weighting parameter from Equation 1; different colored curves are shown for different $\lambda$ values in the other layer.

### 4.5.3 Invariance Tests

We quantify invariance encoded in the unsupervised learned features with invariance tests. In this experiment, we take the approach described in [4] and measure the change in features as input image undergoes transformations. A patch is extracted from a natural image, and transformed through translation, rotation and zoom. We measure the Mean Squared Error (MSE) between the $L_2$ normalized feature vector of the transformed patch and the feature vector of the original patch [3]. The normalized MSE is plotted against the amount of translation, rotation, and zoom. Results of invariance tests are

shown in Figure 8[4]. In these plots, lower curves indicates higher levels of invariance. Our features trained with temporal slowness have better invariance properties compared to features learned only using sparity, and SIFT [5]. Further, simulation of fixation with feature detection and tracking has a visible effect on feature invariance. Specifically, as shown on the left of Figure 8, feature tracking reduces translation invariance in agreement with our analysis in Section 4.5.1. At the same time, middle and right plots of Figure 8 show that feature tracking increases the non-trivial rotation and zoom invariance in the second layer of our temporal slowness features.

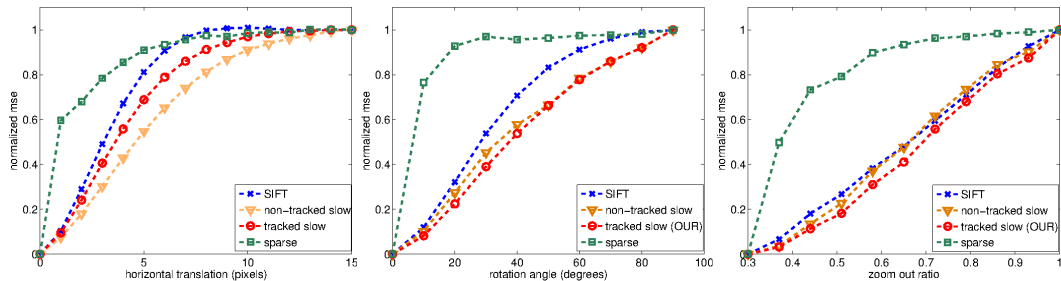

Figure 8: Invariance tests comparing our temporal slowness features using tracked and non-tracked sequences, against SIFT and features trained only with sparsity, shown for different transformations: Translation (left), Rotation (middle) and Zoom (right).

## 5   Conclusion

We have described an unsupervised learning algorithm for learning invariant features from video using the temporal slowness principle. The system is improved by using simulated fixations and smooth pursuit to generate the video sequences provided to the learning algorithm. We illustrate by virtual of visualization and invariance tests, that the learned features are invariant to a collection of non-trivial transformations. With concrete recognition experiments, we show that the features learned from natural videos not only apply to still images, but also give competitive results on a number of object recognition benchmarks. Since our features can be extracted using a feed-forward neural network, they are also easy to use and efficient to compute.

## Footnotes

[1]*http://ai.stanford.edu/ wzou/slow/first_layer_invariance.avi*

[2]http://ai.stanford.edu/ wzou/slow/second_layer_invariance.avi

[3]MSE is normalized against feature dimensions, and averaged across 100 randomly sampled patches. Since the largest distortion makes almost a completely uncorrelated patch, for all features, MSE is normalized against the value at the largest distortion.

## References

[1] N. Li and J. J. DiCarlo. Unsupervised natural experience rapidly alters invariant object representation in visual cortex. *Science*, 2008.

[2] A. Hyvarinen and P. Hoyer. Topographic independent component analysis as a model of v1 organization and receptive fields. *Neural Computation*, 2001.

[3] J.H. van Hateren and D.L. Ruderman. Independent component filters of natural images compared with simple cells in primary visual cortex. *Proc Royal Society*, 1998.

[4] K. Kavukcuoglu, M. Ranzato, R. Fergus, and Y. LeCun. Learning invariant features through topographic filter maps. In *CVPR*, 2009.

[5] D. Cox, P. Meier, N. Oertelt, and J. DiCarlo. 'Breaking' position-invariant object recognition. *Nature Neuroscience*, 2005.

[6] T. Masquelier and S.J. Thorpe. Unsupervised learning of visual features through spike timing dependent plasticity. *PLoS Computational Biology*, 2007.

[7] P. Berkes and L. Wiskott. Slow feature analysis yields a rich repertoire of complex cell properties. *Journal of Vision*, 2005.

[8] E. P. Simoncelli S. Lyu. Nonlinear image representation using divisive normalization. In *CVPR*, 2008.

[9] J. P. Lewis. Fast normalized cross-correlation. In *Vision Interface*, 1995.

[10] A. Hyvarinen, J. Hurri, and J. Vayrynen. Bubbles: a unifying framework for low-level statistical properties of natural image sequences. *Optical Society of America*, 2003.

---

[4]Translation test is performed with 16x16 patches and first layer features, rotation and zoom tests are performed with 32x32 patches and second layer features.

[5]We use SIFT in the VLFeat toolbox [41] *http://www.vlfeat.org/*

[11] J. Hurri and A. Hyvarinen. Temporal coherence, natural image sequences and the visual cortex. In *NIPS*, 2006.

[12] J. Bergstra and Y. Bengio. Slow, decorrelated features for pretraining complex cell-like networks. In *NIPS*, 2009.

[13] R. Raina, A. Madhavan, and A. Y. Ng. Large-scale deep unsupervised learning using graphics processors. In *ICML*, 2009.

[14] A. Coates, H. Lee, and A. Y. Ng. An analysis of single layer networks in unsupervised feature learning. In *AISTATS*, 2011.

[15] B.A. Olshausen and D.J. Field. How close are we to understanding v1? *Neural Computation*, 2005.

[16] A. Coates and A. Ng. The importance of encoding versus training with sparse coding and vector quantization. In *ICML*, 2011.

[17] Q. V. Le, J. Ngiam, Z. Chen, D. Chia, P. W. Koh, and A. Y. Ng. Tiled convolutional neural networks. In *Advances in Neural Information Processing Systems*, 2010.

[18] Q. V. Le, M. A. Ranzato, R. Monga, M. Devin, K. Chen, G. S. Corrado, J. Dean, and A. Y. Ng. Building high-level features using large scale unsupervised learning. In *ICML*, 2012.

[19] A. Coates and A. Y. Ng. Selecting receptive fields in deep networks. In *NIPS*, 2011.

[20] H. Mobahi, R. Collobert, and Jason Weston. Deep learning from temporal coherence in video. In *ICML*, 2009.

[21] M. Franzius, N. Wilbert, and L. Wiskott. Invariant object recognition with Slow Feature Analysis. In *ICANN*, 2008.

[22] B. Olshausen, C. Cadieu, J. Culpepper, and D.K. Warland. Bilinear models of natural images. In *Proc. SPIE 6492*, 2007.

[23] R. P. N. Rao D. B. Grimes. Bilinear sparse coding for invariant vision.

[24] C. Cadieu and B. Olshausen. Learning tranformational invariants from natural movies. In *NIPS*, 2009.

[25] S. Thrun D. Stavens. Unsupervised learning of invariant features using video. In *CVPR*, 2010.

[26] C. Leistner, M. Godec, S. Schulter, M. Werlberger, A. Saffari, and H. Bischof. Improving classifiers with unlabeled weakly-related videos. In *CVPR*, 2011.

[27] T. Lee and S. Soatto. Video-based descriptors for object recognition. *Image and Vision Computing*, 2011.

[28] D. E. Rumelhart, G. E. Hinton, and R. J. Williams. Learning representations by back-propagating errors. *Nature*, 1986.

[29] Y. Bengio and Y. LeCun. Scaling learning algorithms towards AI. In *Large-Scale Kernel Machines*, 2007.

[30] A. Hyvarinen, J. Hurri, and P.O. Hoyer. *Natural Image Statistics*. Springer, 2009.

[31] Q. V. Le, A. Karpenko, J. Ngiam, and A. Y. Ng. ICA with reconstruction cost for efficient overcomplete feature learning. In *NIPS*, 2011.

[32] H. Wersing and E. Kröner. Learning optimized features for hierarchical models of invariant object recognition. *Neural Computation*, 2003.

[33] L. Fei-Fei, R. Fergus, and P. Perona. Learning generative visual models from few training examples: an incremental bayesian approach tested on 101 object categories.

[34] A. Coates, H. Lee, and A. Ng. An analysis of single-layer networks in unsupervised feature learning. In *AISTATS 14*, 2010.

[35] N. Kumar, A. C. Berg, P. N. Belhumeur, and S. K. Nayar. Attribute and simile classifiers for face verification. In *ICCV*, 2009.

[36] K. Kavukcuoglu, P. Sermanet, Y. Boureau, K. Gregor, M. Mathieu, and Y. LeCun. Learning convolutional feature hierarchies for visual recognition. In *NIPS*, 2010.

[37] J. Yang, K. Yu, Y. Gong, and T. Huang. Linear spatial pyramid matching using sparse coding for image classification. In *CVPR*, 2009.

[38] K. Yu, Y. Lin, and J. Lafferty. Learning image representations from the pixel level via hierarchical sparse coding. In *CVPR*, 2011.

[39] Y-Lan Boureau, Francis Bach, Yann LeCun, and Jean Ponce. Learning mid-level features for recognition. In *CVPR*, 2010.

[40] J. Ngiam, P. W. Koh, Z. Chen, S. Bhaskar, and A. Y. Ng. Sparse filtering. In *NIPS*, 2011.

[41] A. Vedaldi and B. Fulkerson. VLFeat: An open and portable library of computer vision algorithms, 2008.

